# Hierarchically Supervised Latent Dirichlet Allocation

**Adler Perotte**    **Nicholas Bartlett**    **Noémie Elhadad**    **Frank Wood**
Columbia University, New York, NY 10027, USA
{ajp9009@dbmi,bartlett@stat,noemie@dbmi,fwood@stat}.columbia.edu

## Abstract

We introduce hierarchically supervised latent Dirichlet allocation (HSLDA), a model for hierarchically and multiply labeled bag-of-word data. Examples of such data include web pages and their placement in directories, product descriptions and associated categories from product hierarchies, and free-text clinical records and their assigned diagnosis codes. Out-of-sample label prediction is the primary goal of this work, but improved lower-dimensional representations of the bag-of-word data are also of interest. We demonstrate HSLDA on large-scale data from clinical document labeling and retail product categorization tasks. We show that leveraging the structure from hierarchical labels improves out-of-sample label prediction substantially when compared to models that do not.

## 1  Introduction

There exist many sources of unstructured data that have been partially or completely categorized by human editors. In this paper we focus on unstructured text data that has been, at least in part, manually categorized. Examples include but are not limited to webpages and curated hierarchical directories of the same [1], product descriptions and catalogs, and patient records and diagnosis codes assigned to them for bookkeeping and insurance purposes. In this work we show how to combine these two sources of information using a single model that allows one to categorize new text documents automatically, suggest labels that might be inaccurate, compute improved similarities between documents for information retrieval purposes, and more. The models and techniques that we develop in this paper are applicable to other data as well, namely, any unstructured representations of data that have been hierarchically classified (e.g., image catalogs with bag-of-feature representations).

There are several challenges entailed in incorporating a hierarchy of labels into the model. Among them, given a large set of potential labels (often thousands), each instance has only a small number of labels associated to it. Furthermore, there are no naturally occurring negative labeling in the data, and the absence of a label cannot always be interpreted as a negative labeling.

Our work operates within the framework of topic modeling. Our approach learns topic models of the underlying data and labeling strategies in a joint model, while leveraging the hierarchical structure of the labels. For the sake of simplicity, we focus on "is-a" hierarchies, but the model can be applied to other structured label spaces. We extend supervised latent Dirichlet allocation (sLDA) [6] to take advantage of hierarchical supervision. We propose an efficient way to incorporate hierarchical information into the model. We hypothesize that the context of labels within the hierarchy provides valuable information about labeling.

We demonstrate our model on large, real-world datasets in the clinical and web retail domains. We observe that hierarchical information is valuable when incorporated into the learning and improves our primary goal of multi-label classification. Our results show that a joint, hierarchical model outperforms a classification with unstructured labels as well as a disjoint model, where the topic model and the hierarchical classification are inferred independently of each other.

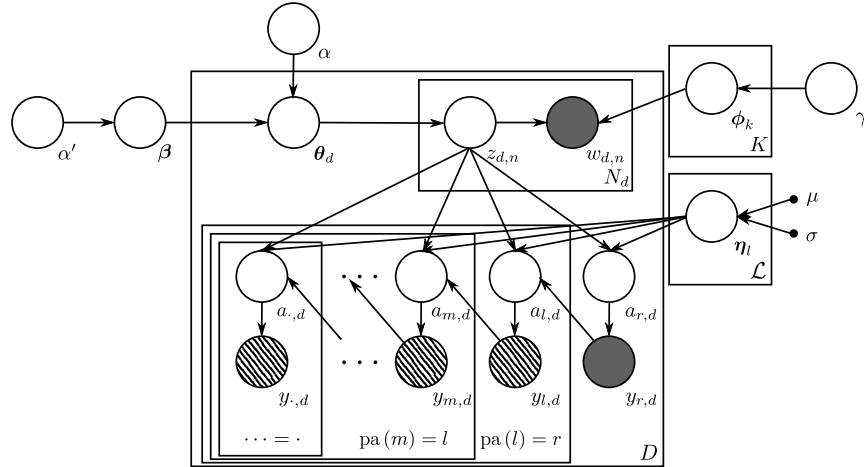

Figure 1: HSLDA graphical model

The remainder of this paper is as follows. Section 2 introduces hierarchically supervised LDA (HSLDA), while Section 3 details a sampling approach to inference in HSLDA. Section 4 reviews related work, and Section 5 shows results from applying HSLDA to health care and web retail data.

## 2  Model

HSLDA is a model for hierarchically, multiply-labeled, bag-of-word data. We will refer to individual groups of bag-of-word data as documents. Let $w_{n,d} \in \Sigma$ be the $n$th observation in the $d$th document. Let $\mathbf{w}_d = \{w_{1,d}, \ldots, w_{1,N_d}\}$ be the set of $N_d$ observations in document $d$. Let there be $D$ such documents and let the size of the vocabulary be $V = |\Sigma|$. Let the set of labels be $\mathcal{L} = \{l_1, l_2, \ldots, l_{|\mathcal{L}|}\}$. Each label $l \in \mathcal{L}$, except the root, has a parent $\mathrm{pa}(l) \in \mathcal{L}$ also in the set of labels. We will for exposition purposes assume that this label set has hard "is-a" parent-child constraints (explained later), although this assumption can be relaxed at the cost of more computationally complex inference. Such a label hierarchy forms a multiply rooted tree. Without loss of generality we will consider a tree with a single root $r \in \mathcal{L}$. Each document has a variable $y_{l,d} \in \{-1, 1\}$ for every label which indicates whether the label is applied to document $d$ or not. In most cases $y_{i,d}$ will be unobserved, in some cases we will be able to fix its value because of constraints on the label hierarchy, and in the relatively minor remainder its value will be observed. In the applications we consider, only positive labels are observed.

The constraints imposed by an is-a label hierarchy are that if the $l$th label is applied to document $d$, i.e., $y_{l,d} = 1$, then all labels in the label hierarchy up to the root are also applied to document $d$, i.e., $y_{\mathrm{pa}(l),d} = 1, y_{\mathrm{pa}(\mathrm{pa}(l)),d} = 1, \ldots, y_{r,d} = 1$. Conversely, if a label $l'$ is marked as not applying to a document then no descendant of that label may be applied to the same. We assume that at least one label is applied to every document. This is illustrated in Figure 1 where the root label is always applied but only some of the descendant labelings are observed as having been applied (diagonal hashing indicates that potentially some of the plated variables are observed).

In HSLDA, documents are modeled using the LDA mixed-membership mixture model with global topic estimation. Label responses are generated using a conditional hierarchy of probit regressors. The HSLDA graphical model is given in Figure 1. In the model, $K$ is the number of LDA "topics" (distributions over the elements of $\Sigma$), $\boldsymbol{\phi}_k$ is a distribution over "words," $\boldsymbol{\theta}_d$ is a document-specific distribution over topics, $\boldsymbol{\beta}$ is a global distribution over topics, $\mathrm{Dir}_K(\cdot)$ is a $K$-dimensional Dirichlet distribution, $\mathcal{N}_K(\cdot)$ is the $K$-dimensional Normal distribution, $\mathbf{I}_K$ is the $K$ dimensional identity matrix, $\mathbf{1}_d$ is the $d$-dimensional vector of all ones, and $\mathbb{I}(\cdot)$ is an indicator function that takes the value 1 if its argument is true and 0 otherwise. The following procedure describes how to generate from the HSLDA generative model.

1. For each topic $k = 1, \ldots, K$
   - Draw a distribution over words $\boldsymbol{\phi}_k \sim \text{Dir}_V(\gamma \mathbf{1}_V)$
2. For each label $l \in \mathcal{L}$
   - Draw a label application coefficient $\boldsymbol{\eta}_l \mid \mu, \sigma \sim \mathcal{N}_K(\mu \mathbf{1}_K, \sigma \mathbf{I}_K)$
3. Draw the global topic proportions $\boldsymbol{\beta} \mid \alpha' \sim \text{Dir}_K(\alpha' \mathbf{1}_K)$
4. For each document $d = 1, \ldots, D$
   - Draw topic proportions $\boldsymbol{\theta}_d \mid \boldsymbol{\beta}, \alpha \sim \text{Dir}_K(\alpha \boldsymbol{\beta})$
   - For $n = 1, \ldots, N_d$
     - Draw topic assignment $z_{n,d} \mid \boldsymbol{\theta}_d \sim \text{Multinomial}(\boldsymbol{\theta}_d)$
     - Draw word $w_{n,d} \mid z_{n,d}, \boldsymbol{\phi}_{1:K} \sim \text{Multinomial}(\boldsymbol{\phi}_{z_{n,d}})$
   - Set $y_{r,d} = 1$
   - For each label $l$ in a breadth first traversal of $\mathcal{L}$ starting at the children of root $r$
     - Draw $a_{l,d} \mid \bar{\mathbf{z}}_d, \boldsymbol{\eta}_l, y_{\text{pa}(l),d} \sim \begin{cases} \mathcal{N}(\bar{\mathbf{z}}_d^T \boldsymbol{\eta}_l, 1), & y_{\text{pa}(l),d} = 1 \\ \mathcal{N}(\bar{\mathbf{z}}_d^T \boldsymbol{\eta}_l, 1) \mathbb{I}(a_{l,d} < 0), & y_{\text{pa}(l),d} = -1 \end{cases}$
     - Apply label $l$ to document $d$ according to $a_{l,d}$

$$y_{l,d} \mid a_{l,d} = \begin{cases} 1 & \text{if } a_{l,d} > 0 \\ -1 & \text{otherwise} \end{cases}$$

Here $\bar{\mathbf{z}}_d^T = [\bar{z}_1, \ldots, \bar{z}_k, \ldots, \bar{z}_K]$ is the empirical topic distribution for document $d$, in which each entry is the percentage of the words in that document that come from topic $k$, $\bar{z}_k = N_d^{-1} \sum_{n=1}^{N_d} \mathbb{I}(z_{n,d} = k)$.

The second half of step 4 is a substantial part of our contribution to the general class of supervised LDA models. Here, each document is labeled generatively using a hierarchy of conditionally dependent probit regressors [14]. For every label $l \in \mathcal{L}$, both the empirical topic distribution for document $d$ and whether or not its parent label was applied (i.e. $\mathbb{I}(y_{\text{pa}(l),d} = 1)$) are used to determine whether or not label $l$ is to be applied to document $d$ as well. Note that label $y_{l,d}$ can only be applied to document $d$ if its parent label $\text{pa}(l)$ is also applied (these expressions are specific to is-a constraints but can be modified to accommodate different constraints). The regression coefficients $\boldsymbol{\eta}_l$ are independent a priori, however, the hierarchical coupling in this model induces a posteriori dependence. The net effect of this is that label predictors deeper in the label hierarchy are able to focus on finding specific, conditional labeling features. We believe this to be a significant source of the empirical label prediction improvement we observe experimentally. We test this hypothesis in Section 5.

Note that the choice of variables $a_{l,d}$ and how they are distributed were driven at least in part by posterior inference efficiency considerations. In particular, choosing probit-style auxiliary variable distributions for the $a_{l,d}$'s yields conditional posterior distributions for both the auxiliary variables (3) and the regression coefficients (2) which are analytic. This simplifies posterior inference substantially.

In the common case where no negative labels are observed (like the example applications we consider in Section 5), the model must be explicitly biased towards generating data that has negative labels in order to keep it from learning to assign all labels to all documents. This is a common problem in modeling unbalanced data. To see how this model can be biased in this way we draw the reader's attention to the $\mu$ parameter and, to a lesser extent, the $\sigma$ parameter above. Because $\bar{\mathbf{z}}_d$ is always positive, setting $\mu$ to a negative value results in a bias towards negative labelings, i.e. for large negative values of $\mu$, all labels become a priori more likely to be negative ($y_{l,d} = -1$). We explore the ability of $\mu$ to bias out-of-sample label prediction performance in Section 5.

## 3 Inference

In this section we provide the conditional distributions required to draw samples from the HSLDA posterior distribution using Gibbs sampling and Markov chain Monte Carlo. Note that, like in collapsed Gibbs samplers for LDA [16], we have analytically marginalized out the parameters $\boldsymbol{\phi}_{1:K}$

and $\boldsymbol{\theta}_{1:D}$ in the following expressions. Let $\mathbf{a}$ be the set of all auxiliary variables, $\mathbf{w}$ the set of all words, $\boldsymbol{\eta}$ the set of all regression coefficients, and $\mathbf{z}\backslash z_{n,d}$ the set $\mathbf{z}$ with element $z_{n,d}$ removed. The conditional posterior distribution of the latent topic indicators is

$$p\left(z_{n,d} = k \mid \mathbf{z}\backslash z_{n,d}, \mathbf{a}, \mathbf{w}, \boldsymbol{\eta}, \alpha, \boldsymbol{\beta}, \gamma\right) \propto$$

$$\left(c_{(\cdot),d}^{k,-(n,d)} + \alpha\boldsymbol{\beta}_k\right) \frac{c_{w_{n,d},(\cdot)}^{k,-(n,d)} + \gamma}{\left(c_{(\cdot),(\cdot)}^{k,-(n,d)} + V\gamma\right)} \prod_{l \in \mathcal{L}_d} \exp\left\{-\frac{\left(\bar{\mathbf{z}}_d^T \boldsymbol{\eta}_l - a_{l,d}\right)^2}{2}\right\} \qquad (1)$$

where $c_{v,d}^{k,-(n,d)}$ is the number of words of type $v$ in document $d$ assigned to topic $k$ omitting the $n$th word of document $d$. The subscript $(\cdot)$'s indicate to sum over the range of the replaced variable, i.e. $c_{w_{n,d},(\cdot)}^{k,-(n,d)} = \sum_d c_{w_{n,d},d}^{k,-(n,d)}$. Here $\mathcal{L}_d$ is the set of labels which are observed for document $d$.

The conditional posterior distribution of the regression coefficients is given by

$$p(\boldsymbol{\eta}_l \mid \mathbf{z}, \mathbf{a}, \sigma) = \mathcal{N}(\hat{\boldsymbol{\mu}}_l, \hat{\boldsymbol{\Sigma}}) \qquad (2)$$

where

$$\hat{\boldsymbol{\mu}}_l = \hat{\boldsymbol{\Sigma}}\left(\mathbf{1}\frac{\mu}{\sigma} + \bar{\mathbf{Z}}^T \mathbf{a}_l\right) \qquad \hat{\boldsymbol{\Sigma}}^{-1} = \mathbf{I}\sigma^{-1} + \bar{\mathbf{Z}}^T \bar{\mathbf{Z}}.$$

Here $\bar{\mathbf{Z}}$ is a $D \times K$ matrix such that row $d$ of $\bar{\mathbf{Z}}$ is $\bar{\mathbf{z}}_d$, and $\mathbf{a}_l = [a_{l,1}, a_{l,2}, \ldots, a_{l,D}]^T$. The simplicity of this conditional distribution follows from the choice of probit regression [4]; the specific form of the update is a standard result from Bayesian normal linear regression [14]. It also is a standard probit regression result that the conditional posterior distribution of $a_{l,d}$ is a truncated normal distribution [4].

$$p\left(a_{l,d} \mid \mathbf{z}, \mathbf{Y}, \boldsymbol{\eta}\right) \propto \begin{cases} \exp\left\{-\frac{1}{2}\left(a_{l,d} - \boldsymbol{\eta}_l^T \bar{\mathbf{z}}_d\right)\right\} \mathbb{I}\left(a_{l,d}y_{l,d} > 0\right) \mathbb{I}(a_{l,d} < 0), & y_{\mathrm{pa}(l),d} = -1 \\ \exp\left\{-\frac{1}{2}\left(a_{l,d} - \boldsymbol{\eta}_l^T \bar{\mathbf{z}}_d\right)\right\} \mathbb{I}\left(a_{l,d}y_{l,d} > 0\right), & y_{\mathrm{pa}(l),d} = 1 \end{cases} \qquad (3)$$

Note that care must be taken to initialize the Gibbs sampler in a valid state.

HSLDA employs a hierarchical Dirichlet prior over topic assignments (i.e., $\boldsymbol{\beta}$ is estimated from data rather than fixed a priori). This has been shown to improve the quality and stability of inferred topics [26]. Sampling $\boldsymbol{\beta}$ is done using the "direct assignment" method of Teh et al. [25]

$$\boldsymbol{\beta} \mid \mathbf{z}, \alpha', \alpha \sim \mathrm{Dir}\left(m_{(\cdot),1} + \alpha', m_{(\cdot),2} + \alpha', \ldots, m_{(\cdot),K} + \alpha'.\right) \qquad (4)$$

Here $m_{d,k}$ are auxiliary variables that are required to sample the posterior distribution of $\boldsymbol{\beta}$. Their conditional posterior distribution is sampled according to

$$p\left(m_{d,k} = m \mid \mathbf{z}, \mathbf{m}_{-(d,k)}, \boldsymbol{\beta}\right) = \frac{\Gamma\left(\alpha\boldsymbol{\beta}_k\right)}{\Gamma\left(\alpha\boldsymbol{\beta}_k + c_{(\cdot),d}^k\right)} s\left(c_{(\cdot),d}^k, m\right) \left(\alpha\boldsymbol{\beta}_k\right)^m \qquad (5)$$

where $s\left(n, m\right)$ represents stirling numbers of the first kind.

The hyperparameters $\alpha$, $\alpha'$, and $\gamma$ are sampled using Metropolis-Hastings.

## 4  Related Work

In this work we extend supervised latent Dirichlet allocation (sLDA) [6] to take advantage of hierarchical supervision. sLDA is latent Dirichlet allocation (LDA) [7] augmented with per document "supervision," often taking the form of a single numerical or categorical label. It has been demonstrated that the signal provided by such supervision can result in better, task-specific document models and can also lead to good label prediction for out-of-sample data [6]. It also has been demonstrated that sLDA has been shown to outperform both LASSO (L1 regularized least squares regression) and LDA followed by least squares regression [6]. sLDA can be applied to data of the type we consider in this paper; however, doing so requires ignoring the hierarchical dependencies amongst the labels. In Section 5 we constrast HSLDA with sLDA applied in this way.

Other models that incorporate LDA and supervision include LabeledLDA [23] and DiscLDA [18]. Various applications of these models to computer vision and document networks have been explored [27, 9] . None of these models, however, leverage dependency structure in the label space.

In other work, researchers have classified documents into a hierarchy (a closely related task) with naive Bayes classifiers and support vector machines. Most of this work has been demonstrated on relatively small datasets, small label spaces, and has focused on single label classification without a model of documents such as LDA [21, 11, 17, 8].

# 5 Experiments

We applied HSLDA to data from two domains: predicting medical diagnosis codes from hospital discharge summaries and predicting product categories from Amazon.com product descriptions.

## 5.1 Data and Pre-Processing

### 5.1.1 Discharge Summaries and ICD-9 Codes

Discharge summaries are authored by clinicians to summarize patient hospitalization courses. The summaries typically contain a record of patient complaints, findings and diagnoses, along with treatment and hospital course. For each hospitalization, trained medical coders review the information in the discharge summary and assign a series of diagnoses codes. Coding follows the ICD-9-CM controlled terminology, an international diagnostic classification for epidemiological, health management, and clinical purposes.[1] The ICD-9 codes are organized in a rooted-tree structure, with each edge representing an is-a relationship between parent and child, such that the parent diagnosis subsumes the child diagnosis. For example, the code for "Pneumonia due to adenovirus" is a child of the code for "Viral pneumonia," where the former is a type of the latter. It is worth noting that the coding can be noisy. Human coders sometimes disagree [3], tend to be more specific than sensitive in their assignments [5], and sometimes make mistakes [13].

The task of automatic ICD-9 coding has been investigated in the clinical domain. Methods range from manual rules to online learning [10, 15, 12]. Other work had leveraged larger datasets and experimented with K-nearest neighbor, Naive Bayes, support vector machines, Bayesian Ridge Regression, as well as simple keyword mappings, all with promising results [19, 24, 22, 20].

Our dataset was gathered from the NewYork-Presbyterian Hospital clinical data warehouse. It consists of 6,000 discharge summaries and their associated ICD-9 codes (7,298 distinct codes overall), representing all the discharges from the hospital in 2009. All included discharge summaries had associated ICD-9 Codes. Summaries have 8.39 associated ICD-9 codes on average (std dev=5.01) and contain an average of 536.57 terms after preprocessing (std dev=300.29). We split our dataset into 5,000 discharge summaries for training and 1,000 for testing.

The text of the discharge summaries was tokenized with NLTK.[2] A fixed vocabulary was formed by taking the top 10,000 tokens with highest document frequency (exclusive of names, places and other identifying numbers). The study was approved by the Institutional Review Board and follows HIPAA (Health Insurance Portability and Accountability Act) privacy guidelines.

### 5.1.2 Product Descriptions and Categorizations

Amazon.com, an online retail store, organizes its catalog of products in a mulitply-rooted hierarchy and provides textual product descriptions for most products. Products can be discovered by users through free-text search and product category exploration. Top-level product categories are displayed on the front page of the website and lower level categories can be discovered by choosing one of the top-level categories. Products can exist in multiple locations in the hierarchy.

In this experiment, we obtained Amazon.com product categorization data from the Stanford Network Analysis Platform (SNAP) dataset [2]. Product descriptions were obtained separately from the Amazon.com website directly. We limited our dataset to the collection of DVDs in the product catalog.

Our dataset contains 15,130 product descriptions for training and 1,000 for testing. The product descriptions are shorter than the discharge summaries (91.89 terms on average, std dev=53.08).

Overall, there are 2,691 unique categories. Products are assigned on average 9.01 categories (std dev=4.91). The vocabulary consists of the most frequent 30,000 words omitting stopwords.

## 5.2 Comparison Models

We evaluated HSLDA along with two closely related models against the two datasets. The comparison models included sLDA with independent regressors (hierarchical constraints on labels ignored) and HSLDA fit by first performing LDA then fitting tree-conditional regressions. These models were chosen to highlight several aspects of HSLDA including performance in the absence of hierarchical constraints, the effect of the combined inference, and regression performance attributable solely to the hierarchical constraints.

sLDA with independent regressors is the most salient comparison model for our work. The distinguishing factor between HSLDA and sLDA is the additional structure imposed on the label space, a distinction that we hypothesized would result in a difference in predictive performance.

There are two components to HSLDA, LDA and a hierarchically constrained response. The second comparison model is HSLDA fit by performing LDA first followed by performing inference over the hierarchically constrained label space. In this comparison model, the separate inference processes do not allow the responses to influence the low dimensional structure inferred by LDA. Combined inference has been shown to improve performance in sLDA [6]. This comparison model examines not the structuring of the label space, but the benefit of combined inference over both the documents and the label space.

For all three models, particular attention was given to the settings of the prior parameters for the regression coefficients. These parameters implement an important form of regularization in HSLDA. In the setting where there are no negative labels, a Gaussian prior over the regression parameters with a negative mean implements a prior belief that missing labels are likely to be negative. Thus, we evaluated model performance for all three models with a range of values for $\mu$, the mean prior parameter for regression coefficients ($\mu \in \{-3, -2.8, -2.6, \ldots, 1\}$).

The number of topics for all models was set to 50, the prior distributions of $p(\alpha)$, $p(\alpha')$, and $p(\gamma)$ were gamma distributed with a shape parameter of 1 and a scale parameters of 1000.

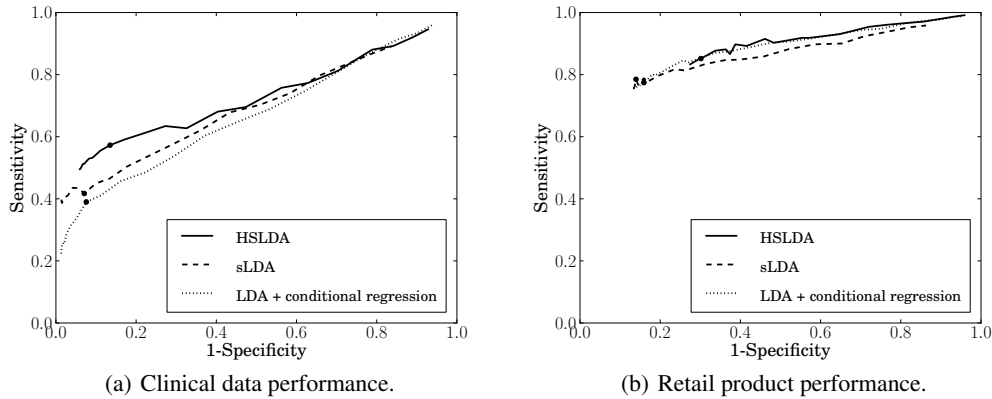

(a) Clinical data performance.     (b) Retail product performance.

Figure 2: ROC curves for out-of-sample label prediction varying $\mu$, the prior mean of the regression parameters. In both figures, solid is HSLDA, dashed are independent regressors + sLDA (hierarchical constraints on labels ignored), and dotted is HSLDA fit by running LDA first then running tree-conditional regressions.

## 5.3 Evaluation and Results

We evaluated our model, HSLDA, against the comparison models with a focus on predictive performance on held-out data. Prediction performance was measured with standard metrics – sensitivity (true positive rate) and 1-specificity (false positive rate).

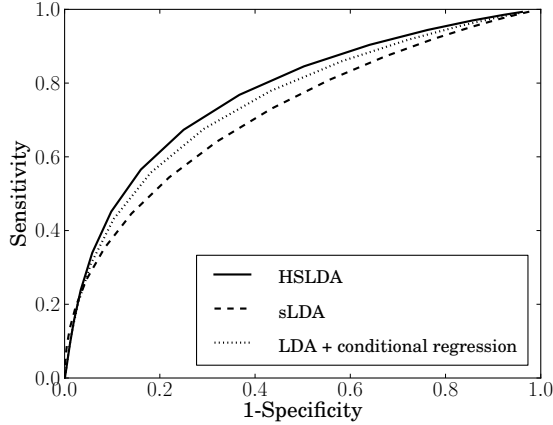

Figure 3: ROC curve for out-of-sample ICD-9 code prediction varying auxiliary variable threshold. $\mu = -1.0$ for all three models in this figure.

The gold standard for comparison was derived from the testing set in each dataset. To make the comparison as fair as possible among models, ancestors of observed nodes in the label hierarchy were ignored, observed nodes were considered positive and descendents of observed nodes were considered to be negative. Note that this is different from our treatment of the observations during inference. Since the sLDA model does not enforce the hierarchical constraints, we establish a more equal footing by considering only the observed labels as being positive, despite the fact that, following the hierarchical constraints, ancestors must also be positive. Such a gold standard will likely inflate the number of false positives because the labels applied to any particular document are usually not as complete as they could be. ICD-9 codes, for instance, lack sensitivity and their use as a gold standard could lead to correctly positive predictions being labeled as false positives [5]. However, given that the label space is often large (as in our examples) it is a moderate assumption that erroneous false positives should not skew results significantly.

Predictive performance in HSLDA is evaluated by $p\left(y_{l,\hat{d}} \mid w_{1:N_{\hat{d}},\hat{d}}, w_{1:N_d,1:D}, y_{l \in \mathcal{L},1:D}\right)$ for each test document, $\hat{d}$. For efficiency, the expectation of this probability distribution was estimated in the following way. Expectations of $\bar{z}_{\hat{d}}$ and $\eta_l$ were estimated with samples from the posterior. Using these expectations, we performed Gibbs sampling over the hierarchy to acquire predictive samples for the documents in the test set. The true positive rate was calculated as the average expected labeling for gold standard positive labels. The false positive rate was calculated as the average expected labeling for gold standard negative labels.

As sensitivity and specificity can always be traded off, we examined sensitivity for a range of values for two different parameters – the prior means for the regression coefficients and the threshold for the auxiliary variables. The goal in this analysis was to evaluate the performance of these models subject to more or less stringent requirements for predicting positive labels. These two parameters have important related functions in the model. The prior mean in combination with the auxiliary variable threshold together encode the strength of the prior belief that unobserved labels are likely to be negative. Effectively, the prior mean applies negative pressure to the predictions and the auxiliary variable threshold determines the cutoff. For each model type, separate models were fit for each value of the prior mean of the regression coefficients. This is a proper Bayesian sensitivity analysis. In contrast, to evaluate predictive performance as a function of the auxiliary variable threshold, a single model was fit for each model type and prediction was evaluated based on predictive samples drawn subject to different auxiliary variable thresholds. These methods are significantly different since the prior mean is varied prior to inference, and the auxiliary variable threshold is varied following inference.

Figure 2(a) demonstrates the performance of the model on the clinical data as an ROC curve varying $\mu$. For instance, a hyperparameter setting of $\mu = -1.6$ yields the following performance: the full HSLDA model had a true positive rate of 0.57 and a false positive rate of 0.13, the sLDA model had

a true positive rate of 0.42 and a false positive rate of 0.07, and the HSLDA model where LDA and the regressions were fit separately had a true positive rate of 0.39 and a false positive rate of 0.08. These points are highlighted in Figure 2(a).

These results indicate that the full HSLDA model predicts more of the the correct labels at a cost of an increase in the number of false positives relative to the comparison models.

Figure 2(b) demonstrates the performance of the model on the retail product data as an ROC curve also varying $\mu$. For instance, a hyperparameter setting of $\mu = -2.2$ yields the following performance: the full HSLDA model had a true positive rate of 0.85 and a false positive rate of 0.30, the sLDA model had a true positive rate of 0.78 and a false positive rate of 0.14, and the HSLDA model where LDA and the regressions were fit separately had a true positive rate of 0.77 and a false positive rate of 0.16. These results follow a similar pattern to the clinical data. These points are highlighted in Figure 2(b).

Figure 3 shows the predictive performance of HSLDA relative to the two comparison models on the clinical dataset as a function of the auxiliary variable threshold. For low values of the auxiliary variable threshold, the models predict labels in a more sensitive and less specific manner, creating the points in the upper right corner of the ROC curve. As the auxiliary variable threshold is increased, the models predict in a less sensitive and more specific manner, creating the points in the lower left hand corner of the ROC curve. HSLDA with full joint inference outperforms sLDA with independent regressors as well as HSLDA with separately trained regression.

## 6    Discussion

The SLDA model family, of which HSLDA is a member, can be understood in two different ways. One way is to see it as a family of topic models that improve on the topic modeling performance of LDA via the inclusion of observed supervision. An alternative, complementary way is to see it as a set of models that can predict labels for bag-of-word data. A large diversity of problems can be expressed as label prediction problems for bag-of-word data. A surprisingly large amount of that kind of data possess structured labels, either hierarchically constrained or otherwise. That HSLDA directly addresses this kind of data is a large part of the motivation for this work. That it outperforms more straightforward approaches should be of interest to practitioners.

Variational Bayes has been the predominant estimation approach applied to sLDA models. Hierarchical probit regression makes for tractable Markov chain Monte Carlo SLDA inference, a benefit that should extend to other sLDA models should probit regression be used for response variable prediction there too.

The results in Figures 2(a) and 2(b) suggest that in most cases it is better to do full joint estimation of HSLDA. An alternative interpretation of the same results is that, if one is more sensitive to the performance gains that result from exploiting the structure of the labels, then one can, in an engineering sense, get nearly as much gain in label prediction performance by first fitting LDA and then fitting a hierarchical probit regression. There are applied settings in which this could be advantageous.

Extensions to this work include unbounded topic cardinality variants and relaxations to different kinds of label structure. Unbounded topic cardinality variants pose interesting inference challenges. Utilizing different kinds of label structure is possible within this framework, but requires relaxing some of the simplifications we made in this paper for expositional purposes.

## Footnotes

[1]http://www.cdc.gov/nchs/icd/icd9cm.htm

[2]http://www.nltk.org

## References

[1] DMOZ open directory project. `http://www.dmoz.org/`, 2002.

[2] Stanford network analysis platform. `http://snap.stanford.edu/`, 2004.

[3] The computational medicine center's 2007 medical natural language processing challenge. http://www.computationalmedicine.org/challenge/previous, 2007.

[4] J. Albert and S. Chib. Bayesian analysis of binary and polychotomous response data. *Journal of the American Statistical Association*, 88(422):669, 1993.

[5] E. Birman-Deych, A. D. Waterman, Y. Yan, D. S. Nilasena, M. J. Radford, and B. F. Gage. Accuracy of ICD-9-CM codes for identifying cardiovascular and stroke risk factors. *Medical Care*, 43(5):480–5, 2005.

[6] D. Blei and J. McAuliffe. Supervised topic models. *Advances in Neural Information Processing*, 20: 121–128, 2008.

[7] D. Blei, A.Y. Ng, and M.I. Jordan. Latent Dirichlet allocation. *J. Mach. Learn. Res.*, 3:993–1022, March 2003. ISSN 1532-4435.

[8] S. Chakrabarti, B. Dom, R. Agrawal, and P. Raghavan. Scalable feature selection, classification and signature generation for organizing large text databases into hierarchical topic taxonomies. *The VLDB Journal*, 7:163–178, August 1998. ISSN 1066-8888.

[9] J. Chang and D. M. Blei. Hierarchical relational models for document networks. *Annals of Applied Statistics*, 4:124–150, 2010. doi: 10.1214/09-AOAS309.

[10] K. Crammer, M. Dredze, K. Ganchev, P.P. Talukdar, and S. Carroll. Automatic code assignment to medical text. *Proceedings of the Workshop on BioNLP 2007: Biological, Translational, and Clinical Language Processing*, pages 129–136, 2007.

[11] S. Dumais and H. Chen. Hierarchical classification of web content. In *Proceedings of the 23rd annual international ACM SIGIR conference on Research and development in information retrieval*, SIGIR '00, pages 256–263, New York, NY, USA, 2000. ACM.

[12] R. Farkas and G. Szarvas. Automatic construction of rule-based ICD-9-CM coding systems. *BMC bioinformatics*, 9(Suppl 3):S10, 2008.

[13] M. Farzandipour, A. Sheikhtaheri, and F. Sadoughi. Effective factors on accuracy of principal diagnosis coding based on international classification of diseases, the 10th revision. *International Journal of Information Management*, 30:78–84, 2010.

[14] A. Gelman, J. B. Carlin, H. S. Stern, and D. B. Rubin. *Bayesian Data Analysis*. Chapman and Hall/CRC, 2nd ed. edition, 2004.

[15] I. Goldstein, A. Arzumtsyan, and Ö. Uzuner. Three approaches to automatic assignment of ICD-9-CM codes to radiology reports. *AMIA Annual Symposium Proceedings*, 2007:279, 2007.

[16] T. L. Griffiths and M. Steyvers. Finding scientific topics. *PNAS*, 101(suppl. 1):5228–5235, 2004.

[17] D. Koller and M. Sahami. Hierarchically classifying documents using very few words. Technical Report 1997-75, Stanford InfoLab, February 1997. Previous number = SIDL-WP-1997-0059.

[18] S. Lacoste-Julien, F. Sha, and M. I. Jordan. DiscLDA: Discriminative learning for dimensionality reduction and classification. In *Neural Information Processing Systems*, pages 897–904.

[19] L. Larkey and B. Croft. Automatic assignment of ICD9 codes to discharge summaries. Technical report, University of Massachussets, 1995.

[20] L. V. Lita, S. Yu, S. Niculescu, and J. Bi. Large scale diagnostic code classification for medical patient records. In *Proceedings of the 3rd International Joint Conference on Natural Language Processing (IJCNLP'08)*, 2008.

[21] A. McCallum, K. Nigam, J. Rennie, and K. Seymore. Building domain-specific search engines with machine learning techniques. In *Proc. AAAI-99 Spring Symposium on Intelligent Agents in Cyberspace*, 1999.

[22] S. Pakhomov, J. Buntrock, and C. Chute. Automating the assignment of diagnosis codes to patient encounters using example-based and machine learning techniques. *Journal of the American Medical Informatics Association (JAMIA)*, 13(5):516–525, 2006.

[23] D. Ramage, D. Hall, R. Nallapati, and C. D. Manning. Labeled LDA: a supervised topic model for credit attribution in multi-labeled corpora. In *Proceedings of the 2009 Conference on Empirical Methods in Natural Language Processing*, pages 248–256, 2009.

[24] B. Ribeiro-Neto, A. Laender, and L. De Lima. An experimental study in automatically categorizing medical documents. *Journal of the American society for Information science and Technology*, 52(5): 391–401, 2001.

[25] Y. W. Teh, M. I. Jordan, M. J. Beal, and D. M. Blei. Hierarchical Dirichlet processes. *Journal of the American Statistical Association*, 101(476):1566–1581, 2006.

[26] H. Wallach, D. Mimno, and A. McCallum. Rethinking LDA: Why priors matter. In Y. Bengio, D. Schuurmans, J. Lafferty, C. K. I. Williams, and A. Culotta, editors, *Advances in Neural Information Processing Systems 22*, pages 1973–1981. 2009.

[27] C. Wang, D. Blei, and L. Fei-Fei. Simultaneous image classification and annotation. In *CVPR*, pages 1903–1910, 2009.

